# MECHANISMS FOR NEUROMODULATION OF BIOLOGICAL NEURAL NETWORKS

**Ronald M. Harris-Warrick**
Section of Neurobiology and Behavior
Cornell University
Ithaca, NY 14853

## ABSTRACT

The pyloric Central Pattern Generator of the crustacean stomatogastric ganglion is a well-defined biological neural network. This 14-neuron network is modulated by many inputs. These inputs reconfigure the network to produce multiple output patterns by three simple mechanisms: 1) determining which cells are active; 2) modulating the synaptic efficacy; 3) changing the intrinsic response properties of individual neurons. The importance of modifiable intrinsic response properties of neurons for network function and modulation is discussed.

## 1  INTRODUCTION

Many neural network models aim to understand how a particular process is accomplished by a unique network in the nervous system. Most studies have aimed at circuits for learning or sensory processing; unfortunately, almost no biological data are available on the actual anatomical structure of neural networks serving these tasks, so the accuracy of the theoretical models is unknown. Much more is known concerning the structure and function of motor circuits generating simple rhythmic movements, especially in simpler invertebrate nervous systems (Getting, 1988). Called Central Pattern Generators (CPGs), these are rather small circuits of relatively well-defined composition. The output of the network is easily measured by monitoring the motor patterns causing movement. Research on cellular interactions in CPGs has shown that simple models of fixed circuitry for fixed outputs are oversimplified. Instead, these neural networks have evolved with maximal flexibility in mind, such that modulatory inputs to the circuit can reconfigure it "on the fly" to generate an almost infinite variety of motor patterns. These modulatory inputs, using slow transmitters such as monoamines and peptides, can change every component of the network, thus constructing multiple functional circuits from a single network (Harris-Warrick, 1988). In this paper, I will describe a model biological system to demonstrate the types of flexibility that are built into real neural networks.

## 2  THE CRUSTACEAN STOMATOGASTRIC GANGLION

The pyloric CPG in the stomatogastric ganglion (STG) of lobsters and crabs is the best-understood neural circuit (Selverston and Moulins, 1987). The STG is a tiny ganglion of 30 neurons that controls rhythmic movements of the foregut. The pyloric CPG controls the peristaltic pumping and filtering movements of the pylorus, or posterior part of the foregut. This network contains 14 neurons, each of which is unambiguously assignable to one of 6 cell types (Figure 1A). Since each neuron can be identified from preparation to preparation, detailed studies of the properties of each cell are possible. Thanks to the careful work of Selverston and Marder and their colleagues, the anatomical synaptic circuitry is completely known (Fig.1A), and consists of chemical synaptic inhibition and electrotonic coupling; there is no chemical excitation in the circuit (Miller,1987).

Despite the complete knowledge of the synaptic connections within this network, the major question of "how it works" is still an important topic of neurobiological research. Early modelling efforts (summarized in Hartline, 1987) showed that, while the pattern of mutual synaptic inhibition provided important insights into the phase relations of the neurons active in the three-phase motor pattern, pure connectionist models with simple threshold elements for neurons were insufficient to explain the motor pattern generated by the network. It has been necessary to understand the intrinsic response properties of each neuron in the circuit, which differ markedly from one another in their responses to identical stimuli. Most importantly, as will be described below, all 14 neurons are conditional oscillators, capable (under the appropriate conditions) of generating rhythmic bursts of action potentials in the absence of synaptic input (Bal et al, 1988). This and other intrinsic properties of the neurons, coupled with the pattern of mutual synaptic inhibition within the circuitry, has generated relatively good models of the pyloric motor pattern under a specified set of conditions (Hartline, 1987).

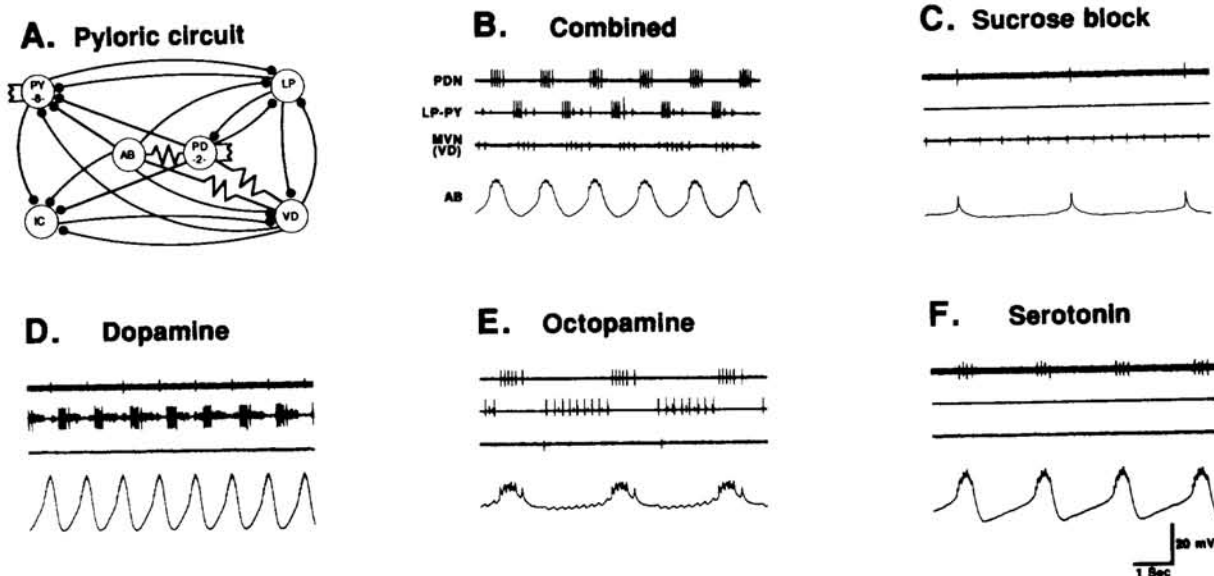

Figure 1: Multiple motor patterns from the pyloric network in the presence of different neurotransmitters. **A.** Synaptic wiring diagram of the pyloric CPG. **B.-F.** Motor patterns observed under different conditions (see text). PDN,LP-PY,MVN traces: extracellular recordings of action potentials from indicated neurons. AB: intracellular recording from the AB interneuron. From Harris-Warrick and Flamm (1987a).

## 3  MULTIPLE MOTOR PATTERNS PRODUCED BY AN ANATOMICALLY FIXED NEURAL NETWORK

When the STG is dissected with intact inputs from other ganglia, the pyloric CPG generates a stereotyped motor pattern (Miller,1987). However, *in vivo,* the network generates a widely varying motor pattern, depending on the feeding state of the animal (Rezer and Moulins, 1983). The motor pattern varies in the cycle frequency and regularity, which cells are active, the intensity of cell firing, and phase relations.

This variability can be mimicked *in vitro,* where experimental control over the system is better. Two major experimental approaches have been used. First, transmitters and modulators that are present in the input nerve to the STG can be bath-applied, producing unique variants on the basic motor theme. Second, identified modulatory neurons can be selectively stimulated, activating and altering the ongoing motor pattern.

As an example, the effects of the monoamines dopamine (DA), serotonin (5HT) and octopamine (OCT) on the pyloric motor pattern are shown in Figure 1. When modulatory inputs from other ganglia are present, the pyloric rhythm cycles strongly, with all neurons active (Combined). Removal of these inputs usually causes the rhythm to cease, and cells are either silent or fire tonically (Sucrose Block). Bath application of some of the transmitters present in the input nerve can restore rhythmic cycling. However, the motor pattern induced is different and unique for each transmitter tested: clearly the patterns induced by DA, 5HT and OCT differ markedly in frequency, intensity, active cells and phasing (Flamm and Harris-Warrick, 1986a). The conclusion is that an anatomically fixed network can generate a variety of outputs in the presence of different modulatory inputs: the anatomy of the network does not determine its output.

## 4 MECHANISMS FOR ALTERATION OF NEURAL NETWORK OUTPUT BY NEUROMODULATORS

We have studied the cellular mechanisms used by monoamines to modify the pyloric rhythm. To do this, we isolate a single neuron or single synaptic interaction by selective killing of other neurons or pharmacological blockade of synapses (Flamm and Harris-Warrick, 1986b). The amine is then added and its direct effects on the neuron or synapse determined. Nearly every neuron in the network responded directly to all three amines we tested. However, even in this simple 14-neuron circuit, different neurons responded differently to a single amine. For example, DA induced rhythmic oscillations and bursting in one cell type, hyperpolarized and silenced two others, and depolarized the remaining cells to fire tonically (Fig.2). Thus, one cannot use the knowledge of the effects of a transmitter on one neuron to infer its actions on other neurons in the same circuit.Our studies of the actions of DA, 5HT and OCT on the pyloric network have demonstrated three simple mechanisms for altering the output from a network.

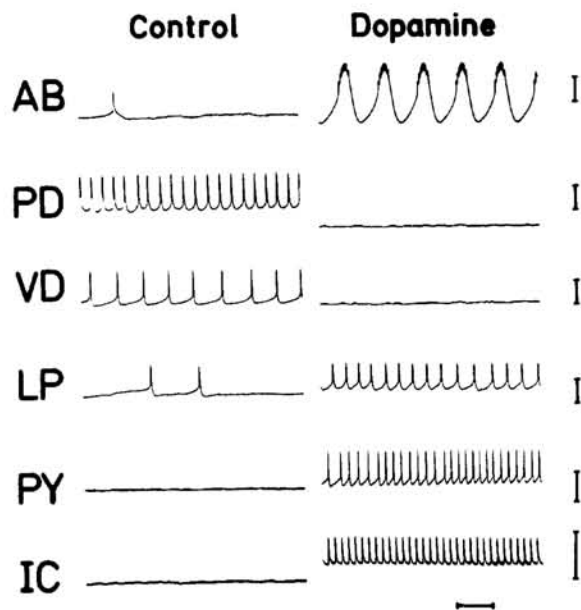

**Figure 2:** Actions of dopamine on isolated neurons from the pyloric network. **Control:** Activity of each neuron when totally isolated from all synaptic input. **Dopamine:** Activity of isolated cell during bath application of $10^{-4}$M dopamine.

## 4.1  ALTERATION OF THE NEURONS THAT ARE ACTIVE PARTICIPANTS IN THE FUNCTIONAL CIRCUIT

By simply exciting a silent cell or inhibiting an active cell, a neuromodulator can determine which of the cells in a network will actively participate in the generation of the motor pattern. Some cells thus are physiologically inactive, even though they are anatomically present.

However, in some cases, unaffected cells can make a significant contribution to the motor pattern. Hooper and Marder (1986) have shown that the peptide proctolin activates the pyloric rhythm and induces rhythmic oscillations in one neuron. Proctolin has no effect on three other neurons that are electrically coupled to the oscillating neuron; these cells impose an electrical drag on the oscillator neuron, causing it to cycle more slowly than it does when isolated from these cells. Thus, the unaffected cells cause the whole motor pattern to cycle more slowly.

## 4.2 ALTERATION OF THE SYNAPTIC EFFICACY OF CONNECTIONS WITHIN THE NETWORK

The flexibility of synaptic interactions is well-known and is used in virtually all models of plasticity in neural networks. By changing the amount of transmitter released from the pre-synaptic terminal or the post-synaptic responsiveness (either by altering the membrane resistance or the number of receptors), the strength of a synapse can be altered over an order of magnitude. Obviously, this will have important effects on the phase relations of neurons firing in the network.

In the STG, the situation is complicated by the fact that graded synapses are the primary form of chemical communication: the cells release transmitter as a continuous function of membrane potential, and do not require action potentials to trigger release (Graubard,

1978).  Some neurons even release transmitter at rest and must be hyperpolarized to block release.  We have shown that graded synaptic transmission is also strongly modulated by monoamines, which can completely eliminate some synapses while strengthening others (Fig.3; Johnson and Harris-Warrick, 1990).  Amines can change the apparent threshold for transmitter release or the functional strength of the synapse.  Modulation of graded transmission thus allows delicate adjustments of the phasing between cells in the motorpattern, which is often determined by synaptic interactions.  Graded synaptic transmission occurs in many species, so this could turn out to be a general form of plasticity.

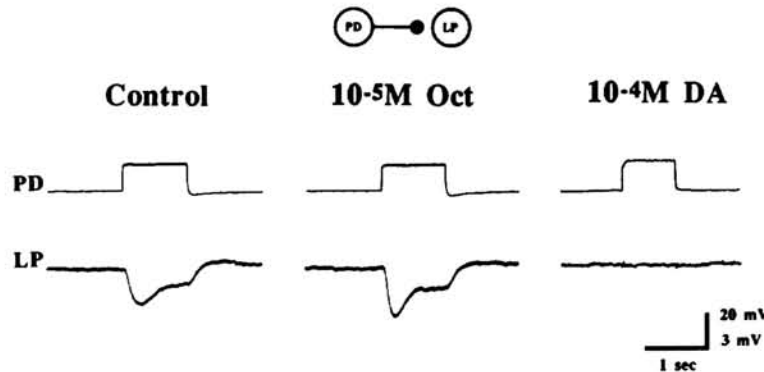

**Figure 3:** Modulation of graded synaptic transmission from the PD neuron to the LP neuron by octopamine and dopamine.  Experiment done in the presence of tetrodotoxin to abolish action potentials.  Other synaptic inputs to these cells have been eliminated.

In one case, modulation of graded transmission results in a sign reversal of the synaptic interaction between two cells (Johnson and Harris-Warrick, 1990).  In the pyloric CPG, the PD neurons weakly inhibit the IC neuron by a graded chemical mechanism, but in addition the two cells are weakly electrically coupled.  This mixed synapse is weak and variable.  Dopamine weakens the chemical inhibition: the electrical coupling dominates and the IC cell depolarizes upon PD depolarization.  Octopamine strengthens the chemical inhibition, and the IC cell hyperpolarizes upon PD depolarization.  Combined chemical and electrical synaptic interactions have been detected in many other preparations, and thus can underly flexibility in the strength and sign of synaptic interactions.

## 4.3    ALTERATION OF THE INTRINSIC RESPONSE PROPERTIES OF THE NETWORK NEURONS

The physiological response properties of neurons within a network are not fixed, but can be extensively altered by neuromodulators.  As a consequence, the response to an identical synaptic input can vary radically in the presence of different neuromodulators.

### 4.3.1    Induction of bistable firing properties

Many neurons in both vertebrates and invertebrates are capable of firing in "plateau potentials", where a brief excitatory stimulus triggers a prolonged depolarized plateau, with tonic spiking for many seconds, which can be prematurely truncated by a brief hyperpolarizing input (Hartline et al, 1988).  Thus, the neuron shows bistable properties: brief synaptic inputs can step it between two relatively stable resting potentials which differ markedly in spike frequency.  This property is plastic, and can be induced or

suppressed by neuromodulatory inputs. For example, Fig. 4 shows the DG neuron in the STG. Under control conditions, a brief depolarizing current injection causes a small depolarization that is subthreshold for spike initiation. However, after stimulating a serotonergic/cholinergic modulatory neuron (called GPR), the same brief current injection induces a prolonged burst of spikes on a depolarized plateau potential (Katz and Harris-Warrick, 1989). Similar results have been obtained in turtle and cat spinal motor neurons after application of monoamines such as serotonin or its biochemical precursor (Hounsgaard et al,1988; Hounsgaard and Kiehn,1989). Stimulation of a modulatory neuron can also disable the plateau potentials that are normally present in a neuron (Nagy et al, 1988).

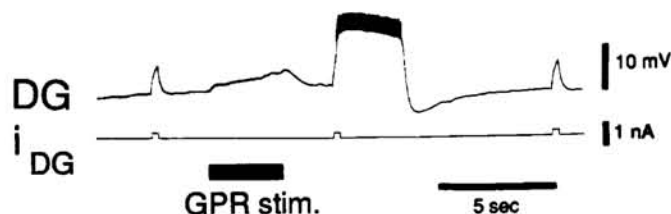

**Figure 4**: Induction of plateau potential capability in DG neuron by stimulation of a serotonergic/cholinergic sensory neuron, GPR.

### 4.3.2 Induction of endogenous rhythmic bursting

A more extreme form of modulation can occur where the modulatory stimulus induces endogenous rhythmic oscillations in membrane potential underlying rhythmic bursts of action potentials. For example, in Figure 4, the pyloric AB neuron shows no intrinsic oscillatory capabilities when it is isolated from all synaptic input. Bath application of monoamines such as DA, 5HT and OCT induce rhythmic bursting in this isolated cell (Flamm and Harris-Warrick, 1986b). Brief stimulation of the serotonergic/cholinergic GPR neuron can also induce or enhance rhythmic bursting that outlasts the stimulus by

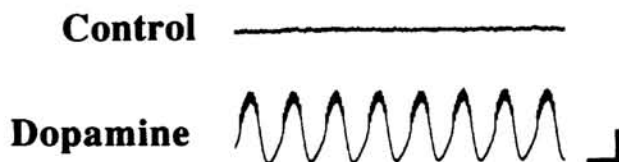

**Figure 5**: Induction of rhythmic bursting in a synaptically isolated AB neuron by bath application of dopamine ($10^{-4}$M).

several minutes. The quantitative details of the bursting (cycle frequency, oscillation amplitude, spike frequency, etc.) are different with each amine, due to different ionic mechanisms for burst generation (Harris-Warrick and Flamm, 1987b). Since the AB neuron is the major pacemaker in the pyloric CPG, these differences underly the marked differences in pyloric rhythm frequency seen with the amines in Fig.1. Induction of rhythmic bursting by neuromodulators has been observed in vertebrates (for example, Dekin et al,1985), and this is likely to be a general mechanism.

### 4.3.3    Modulation of post-inhibitory rebound

Most neurons show post-inhibitory rebound, a period of increased excitability following strong inhibition. This is probably due in part to the activation of prolonged inward currents during hyperpolarization (Angstadt and Calabrese, 1989). This property can be modified by biochemical second messengers used by neuromodulators. For example, elevation of cAMP by forskolin enhances post-inhibitory rebound in the pyloric LP neuron (Figure 5; Flamm et al, 1987). As a consequence of this modulation, the cell's response to a simple inhibitory input is radically changed to a biphasic response, with an initial inhibition followed by delayed excitation.

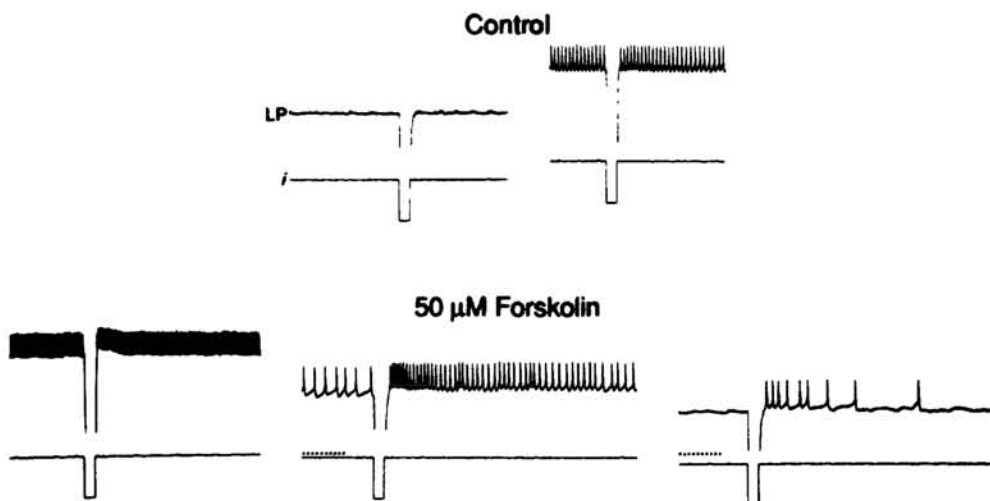

**Figure 6:** Induction of post-inhibitory rebound by forskolin, which elevates cAMP levels, in the LP neuron. **Control**: Hyperpolarizing current injection does not induce post-inhibitory rebound, measured at two different resting potentials. **Forskolin**: Elevation of cAMP depolarizes LP and induces tonic spiking (left). At all membrane potentials, a hyperpolarizing pulse is followed by an enhanced burst of action potentials.

## 5 ENDOGENOUS RELEASE OF NEUROMODULATORS FROM IDENTIFIED NEURONS

Most of the results I have described were obtained with bath application of amines or peptides, a method that can be criticized as being non-physiological. To test this, a number of neurons containing identified neuromodulators have been found, and the action of the naturally released and bath-applied modulator directly compared. An immediate complication arose from these studies: the majority of the known modulatory neurons contain more than one transmitter. All possible combinations have been observed, including a slow transmitter with a fast transmitter, two or more slow transmitters, and multiple fast transmitters. To fully understand the complex changes in network function induced by activity in these neurons, it is necessary to study the actions of all the co-transmitters on all the neurons in the network. This has been recently accomplished in the STG. Here, serotonin is released by a set of sensory cells responding to muscle stretch (Katz et al, 1989). These cells also contain and release acetylcholine (Katz et al,1989). In studying the actions of the two transmitters, remarkable flexibility was uncovered (Katz and Harris-Warrick, 1989,1990). First, not all target neurons responded

to both released transmitters: some responded only to 5HT, while one cell responded only to ACh. Second, the responses to released 5HT were all modulatory, but varied markedly in different cells, mimicking the bath application studies described earlier. Finally, the two transmitters acted over entirely different time scales. ACh induced rapid EPSPs lasting tens to hundreds of msec via nicotinic receptors, while 5HT induced slow prolonged responses lasting many seconds to minutes (for example, Fig.4).

It is now clear that neural networks are targets for multiple neuronal inputs using many different transmitters and modulators. For example, the STG contains only 30 neurons, but is innervated by over 100 axons from other ganglia. Twelve neurotransmitters have thus far been identified in these axons (Marder and Nusbaum,1989), and these are probably a minority of the total that are present. In recordings from the input nerve to the ganglion, many axons are spontaneously active. Thus, the pyloric network is continuously bathed with a varying mixture of transmitters and modulators, allowing for very subtle changes in the firing pattern. *In vivo*, we expect that each modulator plays a small role in the overall mixture that determines the final motor pattern.

# 6  CONCLUSION

The work described here shows conclusively that an anatomically fixed neural network can be modulated to produce a large variety of output patterns. The anatomical connections in the network are necessary but not sufficient to understand the output of the network. Indeed, it is best to think of these networks as libraries of potential components, which are then selected and activated by the modulatory inputs. In addition to altering which neurons are active and altering the synaptic strength in the circuits, I have emphasized the important role of modulation of the intrinsic response properties of the network neurons in determining the final pattern of output. Indeed, if this aspect of modulation is ignored, predictions of the actions of modulators on the final motor pattern are grossly in error.

Many modellers claim that this emphasis on the intrinsic computational properties of single neurons is unique to the invertebrates, which have few cells to work with. In the vertebrates, they argue, the enormous increase in numbers of cells changes the computational rules such that each cell is a simple threshold element, and complex transformations only take place with changes in synaptic efficacy in the circuits. There are absolutely no data to support this hypothesis of "simple cells" in vertebrates. In fact, a great deal of careful work has shown that vertebrate neurons are dynamic elements that show all the complex intrinsic response properties of invertebrate neurons (Llinás,1988). These properties can be changed by neuromodulators, just as in the crustacean STG, such that vertebrate cells can have radically different physiological "personalities" in the presence of different modulators. Network models which ignore the complex computational properties of single neurons thus do not reflect the richness and variability of biological neural networks of both invertebrates and vertebrates alike.

**Acknowledgments**: Supported by NIH Grant NS17323 and Hatch Act NYC-191410.

# 7  BIBLIOGRAPHY

Angstadt, J.D., Calabrese, R.L. (1989) A hyperpolarization-activated inward current in heart interneurons of the medicinal leech. *J. Neurosci.* 9: 2846-2857.

Bal, T., Nagy, F., Moulins, M. (1988) The pyloric central pattern generator in Crustacea: a set of conditional neuronal oscillators. *J. Comp. Physiol. A* 163: 715-727.

Dekin, M.S., Richerson, G.B., Getting, P.A. (1985) Thyrotropin-releasing hormone induces rhythmic bursting in neurons of the nucleus tractus solitarius. *Science* 229:67-69.

Flamm, R.E., Harris-Warrick, R.M. (1986a) Aminergic modulation in lobster stomatogastric ganglion. I. The effects on motor pattern and activity of neurons within the pyloric circuit. *J. Neurophysiol.* 55: 847-865.

Flamm, R.E., Harris-Warrick, R.M. (1986b) Aminergic modulation in lobster stomatogastric ganglion. II. Target neurons of dopamine, octopamine, and serotonin within the pyloric circuit. *J. Neurophysiol.* 55: 866-881.

Flamm, R.E., Fickbohm, D., Harris-Warrick, R.M. (1987) cAMP elevation modulates physiological activity of pyloric neurons in the lobster stomatogastric ganglion. *J. Neurophysiol.* 58: 1370-1386.

Getting, P.A. (1988). Comparative analysis of invertebrate central pattern generators. in: Cohen, A.H., Rossignol, S., Grillner, S. (eds.), Neural Control of Rhythmic Movements in Vertebrates, John Wiley and Sons, New York, pp. 101-127.

Graubard, K. (1978) Synaptic transmission without action potentials: input-output properties of a non-spiking presynaptic neuron. *J. Neurophysiol.* 41: 1014-1025.

Harris-Warrick, R. M. (1988) Chemical modulation of central pattern generators. in: Cohen, A.H., Rossignol, S., Grillner, S.(eds.) Neural Control of Rhythmic Movements in Vertebrates, John Wiley & Sons, New York. pp 285-331.

Harris-Warrick, R.M., Flamm, R.E. (1987a) Chemical modulation of a small central pattern generator circuit. *Trends in Neurosci.* 9: 432-437.

Harris-Warrick, R.M., Flamm, R. E. (1987b) Multiple mechanisms of bursting in a conditional bursting neuron. *J. Neurosci.* 7: 2113-2128.

Hartline, D.K. (1987) Modeling stomatogastric ganglion. in: Selverston, A.I., Moulins, M. (eds.), The Crustacean Stomatogastric System, Springer-Verlag, Berlin, pp. 181-197.

Hartline, D.K., Russell, D.K., Raper, J.A., Graubard, K. (1988) Special cellular and synaptic mechanisms in motor pattern generation. *Comp. Biochem. Physiol.* 91C:115-131.

Hooper, S.L., Marder, E (1987) Modulation of the lobster pyloric rhythm by the peptide proctolin. *J. Neurosci.* 7:2097-2112.

Hounsgaard, J., Kiehn, O. (1989) Serotonin-induced bistability of turtle motoneurones caused by a nifedipine-sensitive calcium plateau potential. *J. Physiol.* 414:265-282.

Hounsgaard, J., Hultborn, H., Jespersen, B., Kiehn, O. (1988) Bistability of alpha-motoneurones in the decerebrate cat and in the acute spinal cat after intravenous 5-hydroxytryptophan. *J. Physiol.* 405:345-367.

Jan, L.Y., Jan, Y.N. (1982) Peptidergic transmission in sympathetic ganglia of the frog. *J. Physiol.* 327: 219-246.

Johnson, B. R., Harris-Warrick, R.M. (1990) Aminergic modulation of graded synaptic transmission in the lobster stomatogastric ganglion. *J. Neurosci.*, in press.

Katz, P.S., Eigg, M.H., Harris-Warrick, R.M. (1989) Serotonergic/cholinergic muscle receptor cells in the crab stomatogastric nervous system. I. Identification and characterization of the gastropyloric receptor cells. *J. Neurophysiol.* 62: 558-570.

Katz, P.S.,  Harris-Warrick, R.M. (1989) Serotonergic/cholinergic muscle receptor cells in the crab stomatogastric nervous system. II. Rapid nicotinic and prolonged modulatory effects on neurons in the stomatogastric ganglion. *J. Neurophysiol.* 62: 571-581.

Katz, P.S., Harris-Warrick, R. M. (1990) Neuromodulation of the crab pyloric central pattern generator by serotonergic/cholinergic proprioceptive afferents. *J. Neurosci.*, in press.

Llinás, R.R. (1988) The intrinsic electrophysiological properties of mammalian neurons: insights into central nervous function. *Science* 242: 1654-1664.

Marder, E., Nusbaum, M.P. (1989) Peptidergic modulation of the motor pattern generators in the stomatogastric ganglion. in: Carew, T.J., Kelley, D.B. (eds.), Perspectives in Neural Systems and Behavior, Alan R. Liss, Inc., New York. pp 73-91.

Miller, J.P. (1987) Pyloric mechanisms.  in: Selverston, A.I., Moulins, M. (eds.) The Crustacean Stomatogastric System , Springer-Verlag, Berlin, pp. 109-136.

Nagy, F., Dickinson, P.S., Moulins, M.  (1988) Control by an identified modulatory neuron of the sequential expression of plateau properties of, and synaptic inputs to, a neuron in a central pattern generator. *J. Neurosci.* 8:2875-2886.

Rezer, E., Moulins, M. (1983) Expression of the crustacean pyloric pattern generator in the intact animal. *J. Comp. Physiol.* 153:17-28.

Selverston, A.I., Moulins, M. (eds.) (1987) The Crustacean Stomatogastric System Springer-Verlag, Berlin, 338 pp.